# Comparison of three classification techniques, CART, C4.5 and Multi-Layer Perceptrons

**A C Tsoi**
Department of Electrical Engineering
University of Queensland
St Lucia, Queensland 4072
Australia

**R A Pearson**
Department of Computer Science
Aust Defence Force Academy
Campbell, ACT 2600
Australia

## Abstract

In this paper, after some introductory remarks into the classification problem as considered in various research communities, and some discussions concerning some of the reasons for ascertaining the performances of the three chosen algorithms, viz., CART (Classification and Regression Tree), C4.5 (one of the more recent versions of a popular induction tree technique known as ID3), and a multi-layer perceptron (MLP), it is proposed to compare the performances of these algorithms under two criteria: classification and generalisation. It is found that, in general, the MLP has better classification and generalisation accuracies compared with the other two algorithms.

## 1 Introduction

Classification of data into categories has been pursued by a number of research communities, viz., applied statistics, knowledge acquisition, neural networks.

In applied statistics, there are a number of techniques, e.g., clustering algorithms (see e.g., Hartigan), CART (Classification and Regression Trees, see e.g., Breiman et al). Clustering algorithms are used when the underlying data naturally fall into a number of groups, the distance among groups are measured by various metrics [Hartigan]. CART [Breiman, et al] has been very popular among applied statisticians. It assumes that the underlying data can be separated into categories, the decision boundaries can either be parallel to the axis or they can be a linear combination of these axes[1]. Under certain assumptions on the input data and their associated

output categories, its properties can be proved rigorously [Breiman et al]. The way in which CART organises its data set is quite sophisticated. For example, it grows a number of decision trees by a cross validation method.

Knowledge acquisition is an important topic in expert systems studies, see e.g., Charniak, McDermott. In this case, one is presented with a subset of input output examples drawn from the set of all possible input output examples exhibited by the underlying system. The problem is how to "distill" a set of rules describing the set of input output examples. The rules are often expressed in the form of "if statement 1, then statement 2, else statement 3". Once this set of rules is obtained, it can be used in a knowledge base for inference or for consulting purposes. It is trivial to observe that the rules can be represented in the form of a binary tree structure. In the process of building this binary tree, the knowledge acquisition system must learn about the set of input output examples. Often this problem is pursued in the machine learning community, see e.g., Michalski et al.

One of the most popular induction tree algorithms is known as ID3, or its later variants, known as C4 (see e.g., Quinlan, Utgoff). There has not been any explicit mention of the underlying assumptions on the data. However, it can be postulated that for an induction tree technqiue to work efficiently, there must be some underlying assumptions on the data set considered. By analogy with CART, it can be observed that an important underlying assumption must be that the data can be divided into categories, the decision boundaries must be parallel to the axes (i.e., it does not find a linear combination of the underlying axes to form a possible decision boundary). In contrast to CART, and similar technqiues, it does not yet have a rigorous theoretical basis. Its learning algorithm, and the way in which it organises the data set are somewhat different from CART.

Recently, there is considerable activities in the study of yet another classification method, known generally as an artificial neural network (ANN) approach (see e.g., Hecht-Nielson). In this approach, the idea is to use a system consisting of artificial neurons with very simple internal dynamics, interconnected to each other for modelling a given set of input output examples. In this approach, one selects an architecture of interconnection of artificial neurons, and a learning algorithm for finding the unknown parameters in the architecture. A particular popular ANN architecture is known as a multi-layer perceptron (MLP). In this architecture, signal travels in only one direction, i.e., there is no feedback from the output to the input. A simple version of this architecture, consisting of only input and output layers of neurons was popularised by Rosenblatt in the 1950's and 1960's. An improved version incorporating possibly more than one layer of hidden layer neurons has been used in the more recent past. A learning algorithm for finding the set of unknown parameters in this architecture while minimising a least square criterion is known as a back propagation algorithm. (see e.g., Rumelhart, McClelland).

There have been much analysis recently in understanding why a MLP can be used in classifying given input output examples, and what underlying assumptions are required (see e.g., Cybenko, Hornik et al). It can be proved that the MLP can be used to approximate any given nonlinear input output mapping given certain not too restrictive assumptions on the mapping, and the underlying input output variables.

Given that the three methods mentioned above, viz., CART, C4.5 (the latest version of the C4 Induction Tree methodology), and MLP, all enjoy popularity in their respective research communities, and that they all perform classification based on a given set of input output examples, a natural question to ask is: how do they perform as compared with one another.

There might be some objections to why a comparison among these algorithms is necessary, since each is designed to operate under some predetermined conditions. Secondly, even if it is shown that a particular algorithm performs better for a set of particular examples, there is no guarantee that the algorithm will perform better under a different set of circumstances. Thus, this may throw some doubt on the desirability of making a comparison among these algorithms.

As indicated above, each algorithm has some underlying assumptions on the construction of a data model, whether these assumptions are made explicit or not. In a practical problem, e.g., power system forecasting [Atlas et al] it is not possible to determine the underlying assumptions in the data. But on an artificially generated example, it is possible to constrain the data so that they would have the desirable characteristics. From this, it is possible to at least make some qualitative statements concerning the algorithms. These qualitative statements may guide a practitioner to watch out for possible pitfalls in applying a particular algorithm to practical problems. Hence, it is worthwhile to carry out comparison studies.

The comparison question is not new. In fact there are already a number of studies carried out to compare the performances of some of or all three algorithms mentioned[2]. For example, Atlas et al compared the performances of CART and MLP. In addition they have considered the performances of these two algorithms to a practical problem, viz., the power system forecasting. Dietterich et al compared the performances of ID3 and MLP, and have applied them to the Text to Speech mapping problem. In general, their conclusions are that the MLP is more accurate in performing generalisation on unseen examples, while the ID3 or CART is much faster in performing the classficiation task.

In this paper, we will consider the performances of all three algorithms, viz., CART, C4.5 and MLP on two criteria:

- Classification capabilities
- Generalisation capabilities

In order to ascertain how these algorithms will perform, we have chosen to study their performances using a closed set of input output examples. In this aspect, we have chosen a version of the Penzias example, first considered by Denker et al. This class of problems has been shown to require at least one hidden layer in a MLP architecture, indicating that the relationship between the input and output is non-linear. Secondly, the problem complexity depends on the number of input neurons (in Cart and C4.5, input features). Hence it is possible to test the algorithms using a progressively complex set of examples.

We have chosen to compare the algorithms under the two critieria because of the

fact that some of them, at least, in the case of CART, were designed for classi-
fication purposes. It was not originally intended for generalisation purposes. By
generalisation, we mean that the trained system is used to predict the categories of
unseen examples when only the input variables are given. The predicted categories
are then compared with the true categories to ascertain how well the trained system
has performed.

The separate comparison is necessary because of the fact that classification and
generalisation are rather different. In classification studies, the main purpose is to
train a system to classify the given set of input output examples. The characteristics
are: good model of the data; good accuracy in classifying the given set of examples.
In generalisation, the main goal is to provide a good accuracy of prediction of output
categories on the set of unseen examples. It does not matter much if the results of
applying the trained data model to the training data set are less accurate.

An important point to note is that all the algorithms have a number of parameters
or procedures which allow them to perform better. For example, it is possible to
vary the a priori assumption on the occurrence of different output categories in
CART, while to perform a similar task in C4.5 or MLP is rather more difficult. It
is possible to train the MLP by ever increasing iterations until the error is small,
given sufficient number of hidden layer neurons. On the other hand, in C4.5, or
CART, the number of iterations is not an externally adjustable parameter.

In order to avoid pitfalls like these, as well as to avoid the criticism of favoring
one algorithm over against another, the results presented here have not consciously
tuned to give the best performance. For example, even though from observations,
we know that the distribution of different output categories is uneven, we have
not made any adjustments to the a priori probabilities in running CART. We will
assume that the output categories occur with equal prior probabilities. We have
not tuned the number of hidden layer neurons in the MLP, except we have taken a
particular number which has been used by others. We have not tuned the learning
rate, nor the momentum rate in the MLP except just a nominal default value which
appears to work for other examples. We have not tuned the C4.5 nor CART apart
from using the default values. Hopefully by doing this, the comparison will appear
fairer.

The structure of the paper is as follows: in section 2, we will describe the classifi-
cation results, while in section 3 we will present generalisation results.

## 2   Comparison of classification performances

Before we present the results of comparing the performances of the algorithms, we
will give a brief description of the testing example used. This example is known as a
clump example in Denker et al, while in Maxwell et al it is refered as the contiguity
example (see [Webb, Lowe]).

There are N input features, each feature can take only the values of 0 or 1. Thus
there are altogether $2^N$ examples. The output class of a particular input feature
vector is the number of clumps involving 1's in the input feature vector. Thus, for
example, if the input feature vector is 00110100, then this is in class 2 as there are
two distinct clumps of 1's in the input features. Hence it is possible to generate

the closed set of all input output examples given a particular value of $N$. For convenience, we will call this an Nth order Penzias example. In our case considered here, we have used $N = 8$, i.e., there are 256 examples in the entire set. The input features are binary equivalent of their ordinal numbers. For example, example 10 is 00001010. This allows us to denote any sample within the set more conveniently. The distribution of the output classes are as follows:

| class | total number |
|-------|--------------|
| 1 | 37 |
| 2 | 126 |
| 3 | 84 |
| 4 | 9 |

For classification purposes, we use all 256 examples as both the training and testing data sets. The following table summarises the classificiation results.

| name | # of errors | accur % |
|------|-------------|---------|
| cart | 96 | 0.625 |
| c4.5 | 105 | 0.59 |
| mlp1 | 117 | 0.54 |
| mlp2 | 47 | 0.82 |

where mlp1 and mlp2 are the values related to the MLP when it has run for 10000 iterations and 100000 iterations respectively. We have run the MLP in the following fashion: we run it 10000 times and then in steps of 10000 iterations but at the beginning of each 10000 iterations it is run with a different initial parameter estimate. In this way, we can ensure that the MLP will not fall into a local minimum. Secondly, we can observe how the MLP accuracies will improve with increasing number of iterations. We found that in general, the MLP converges in about 20000 iterations. After that the number of iterations the results do not improve by a significant amount. In addition, becasue of the way in which we run the experiemnt the convergence would be closer to the average convergence rather than the convergence for a particular initial condition.

The parameter values used in running the experiments are as follows: In the MLP, both the learning rate and the momentum are set at 0.1. The architeture used is: 8 input neurons, 5 hidden layer neurons, and 4 output neurons. In CART, the prior probability is set to be equi-probable. The pruning is performed when the probability of the leaf node is equal 0.5. In C4.5, all the default values are used.

We have also examined the ways in which each algorithm predicts the output categories. We found that none of the algorithms ever predict an output category of 4. This is interesting in that the output category 4 occurs only 9 times out of a total possible of 256. Thus each algorithm, even though it may or may not be able to adjust the prior probability of the output categories, has made an implicit assumption of equal prior probability. This leads to the non occurrence of prediction of category 4 as it is the least frequent occurred one.

Secondly, all algorithms have a default prediction. For example, in CART, the default is class 2, being the most frequently occurred output category in the training examples, while in the case of C4.5, the default is determined by the algorithm. On the other hand, in the cases of CART, or MLP, it is not clear how the default cases

are determined.

Thirdly, the algorithms make mistaken predictions at different places. For example, for sample 1, C4.5 makes the wrong prediction of category 3 while MLP makes the wrong prediction of 2, and CART makes the correct prediction. For sample 9, both CART and C4.5 make a wrong prediction, while MLP makes the correct prediction.

## 3    Comparison of generalisation performances

We have used the same set of input output examples generated by an 8th order Penzias example. For testing the generalisation capabilties, We have used the first 200 examples as the training vector set, and the rest of the vectors in the testing data set.

The results are summarised in the following table:

| name | training | | testing | |
|------|------------|---------|------------|---------|
|      | # of errors | accur % | # of errors | accur % |
| cart | 84 | 0.58 | 34 | 39.3 |
| c4.5 | 97 | 51.5 | 25 | 55.4 |
| mlp1 | 100 | 50 | 28 | 50 |
| mlp2 | 50 | 75 | 25 | 55.4 |

It is noted that the generalisation accuracy of the MLP is better than CART, and is comparable to C4.5.

We have also examined closely the mistakes made by the algorithms as well as the default predictions. In this case, the comments made in section 2 also appear to be true.

## 4    Conclusions

In this paper, we considered three classification algorithms, viz., CART, C4.5, and MLP. We compared their performance both in terms of classification, and generalisation on one example, an 8th order generalised Penzias example. It is found that the MLP once it is converged, in general, has a better classification and generalisation accuracies compared with CART, or C4.5. On the other hand it is also noted that the prediction errors made by each algorithm are different. This indicates that there may be a possibility of combining these algorithms in such a way that their prediction accuracies could be improved. This is presented as a challenge for future research.

## Footnotes

[1] In CART, and C4.5, the axes are the same as the input features

[2]Both Atlas et al, and Diettrich et al were brought to our attention during the conference. Hence some of their conclusions were only communicated to us at that time

## References

J. Hartigan. (1974) *Clustering Algorithms.* J. Wiley, New York.

L. Breiman, J.H. Friedman, R.A. Olshen, J. Stone. (1984) *Classification and Regression Trees.* Wadsworth and Brooks, Monterey, Calif.

E. Charniak, D. McDermott. (1985) *Introduction to Artificial Intelligence.* Ad-

dision Wesley, Reading, Mass.

R. Michalski, J.G. Carbonell, T. Mitchell. (1983) *Machine Learning: An Artificial Intelligence Approach.* Tioga, Palo Alto, Calif.

J.R. Quinlan. (1983) Learning efficient classification procedures and their application to Chess End Games. In R. Michalski et al (ed.), *Machine Learning: An Artificial Intelligence Approach.* Tioga, Palo Alto, Calif.

J. R. Quinlan. (1986) Induction of Decision Trees. *Machine Learning*, 1, 81-106.

P. Utgoff. Incremental Induction of Decision Trees. *Machine Learning*, 4, 161-186.

R. Hecht-Nielson. (1990) *Neurocomputing* Addison Wesley, New York.

F. Rosenblatt. (1962) *Principles of Neurodynamics.* Spartan Books, Washington, DC.

D. Rumelhart, J. McClelland. (1987) *Parallel Distributed Processing: Exploration in the Microstructure of Cognition* Volume 1. MIT Press: Bradford Books.

G. Cybenko. (1989) Approximation by superpositions of sigmoidal function. *Mathematics of Control, Signal, and Systems*, 2:4.

K. Hornik, M. Stinchcombe, H. White. (1989) Multi-layer feedforward networks are universal approximators. *Neural Networks*, 2:5, 359-366.

L. Atlas, R. Cole, Y. Muthusamy, A. Lippman, J. Connor, D. Park, M. El-Sharkawi, R. Marks II. (1990). A Performance Comparison of Trained Multilayer Perceptrons and Trained Classification Trees. *Proc IEEE*, 78:10, 1614-1619.

T. Dietterich, H. Hild, G. Bakiri, (1990), "A Comparison of ID3 and Backpropagation for English Text-to-Speech Mapping", Preprint.

J. Denker, et al. (1987) Large automatic learning, rule extraction, and generalisation. *Complex Systems*, 3 877-922.

T. Maxwell, L. Giles, Y.C. Lee. (1987) Generalisation in neural networks, the contiguity problem. *Proc IEEE 1st Int Conf on Neural Networks*, San Diego, Calif.

A.R. Webb, D. Lowe. (1990) The Optimised internal representation of multilayered classifier networks performs nonlinear discriminant analysis. *Neural Networks* 3:4, 367-376.